# Large-Scale Bayesian Multi-Label Learning via Topic-Based Label Embeddings

**Piyush Rai**[†*]**, Changwei Hu**[*]**, Ricardo Henao**[*]**, Lawrence Carin**[*]
[†]CSE Dept, IIT Kanpur    [*]ECE Dept, Duke University
piyush@cse.iitk.ac.in, {ch237,r.henao,lcarin}@duke.edu

## Abstract

We present a scalable Bayesian multi-label learning model based on learning low-dimensional label embeddings. Our model assumes that each label vector is generated as a weighted combination of a set of *topics* (each topic being a distribution over labels), where the combination weights (i.e., the embeddings) for each label vector are conditioned on the observed feature vector. This construction, coupled with a Bernoulli-Poisson link function for each label of the binary label vector, leads to a model with a computational cost that scales in the number of positive labels in the label matrix. This makes the model particularly appealing for real-world multi-label learning problems where the label matrix is usually very massive but highly sparse. Using a data-augmentation strategy leads to full local conjugacy in our model, facilitating simple and very efficient Gibbs sampling, as well as an Expectation Maximization algorithm for inference. Also, predicting the label vector at test time does not require doing an inference for the label embeddings and can be done in closed form. We report results on several benchmark data sets, comparing our model with various state-of-the art methods.

## 1   Introduction

Multi-label learning refers to the problem setting in which the goal is to assign to an object (e.g., a video, image, or webpage) a *subset* of labels (e.g., tags) from a (possibly very large) *set* of labels. The label assignments of each example can be represented using a binary *label vector*, indicating the presence/absence of each label. Despite a significant amount of prior work, multi-label learning [7, 6] continues to be an active area of research, with a recent surge of interest [1, 25, 18, 13, 10] in designing scalable multi-label learning methods to address the challenges posed by problems such as image/webpage annotation [18], computational advertising [1, 18], medical coding [24], etc., where not only the number of examples and data dimensionality are large but the number of labels can also be massive (several thousands to even millions).

Often, in multi-label learning problems, many of the labels tend to be correlated with each other. To leverage the label correlations and also handle the possibly massive number of labels, a common approach is to reduce the dimensionality of the label space, e.g., by projecting the label vectors to a subspace [10, 25, 21], learning a prediction model in that space, and then projecting back to the original space. However, as the label space dimensionality increases and/or the sparsity in the label matrix becomes more pronounced (i.e., very few ones), and/or if the label matrix is only partially observed, such methods tend to suffer [25] and can also become computationally prohibitive.

To address these issues, we present a scalable, fully Bayesian framework for multi-label learning. Our framework is similar in spirit to the label embedding methods based on reducing the label space dimensionality [10, 21, 25]. However, our framework offers the following key advantages: (1) computational cost of training our model scales in the number of ones in the label matrix, which makes our framework easily scale in cases where the label matrix is massive but sparse; (2) our likelihood model for the binary labels, based on a Bernoulli-Poisson link, more realistically models the extreme sparsity of the label matrix as compared to the commonly employed logistic/probit link; and (3) our model is more interpretable - embeddings naturally correspond to topics where each topic is a distribution over labels. Moreover, at test time, unlike other Bayesian methods [10], we do not need to infer the label embeddings of the test example, thereby leading to faster predictions.

In addition to the modeling flexibility that leads to a robust, interpretrable, and scalable model, our framework enjoys full local conjugacy, which allows us to develop simple Gibbs sampling, as well as an Expectation Maximization (EM) algorithm for the proposed model, both of which are simple to implement in practice (and amenable for parallelization).

## 2 The Model

We assume that the training data are given in the form of $N$ examples represented by a feature matrix $\mathbf{X} \in \mathbb{R}^{D \times N}$, along with their labels in a (possibly *incomplete*) label matrix $\mathbf{Y} \in \{0,1\}^{L \times N}$. The goal is to learn a model that can predict the label vector $\boldsymbol{y}_* \in \{0,1\}^L$ for a test example $\boldsymbol{x}_* \in \mathbb{R}^D$.

We model the binary label vector $\boldsymbol{y}_n$ of the $n^{th}$ example by thresholding a count-valued vector $\boldsymbol{m}_n$

$$\boldsymbol{y}_n = \mathbb{1}(\boldsymbol{m}_n \geq 1) \tag{1}$$

which, for each individual binary label $y_{ln} \in \boldsymbol{y}_n$, $l = 1, \ldots, L$, can also be written as $y_{ln} = \mathbb{1}(m_{ln} \geq 1)$. In Eq. (1), $\boldsymbol{m}_n = [m_{1n}, \ldots, m_{Ln}] \in \mathbb{Z}^L$ denotes a *latent* count vector of size $L$ and is assumed drawn from a Poisson

$$\boldsymbol{m}_n \sim \text{Poisson}(\boldsymbol{\lambda}_n) \tag{2}$$

Eq (2) denotes drawing each component of $\boldsymbol{m}_n$ independently, from a Poisson distribution, with rate equal to the corresponding component of $\boldsymbol{\lambda}_n \in \mathbb{R}_+^L$, which is defined as

$$\boldsymbol{\lambda}_n = \mathbf{V}\boldsymbol{u}_n \tag{3}$$

Here $\mathbf{V} \in \mathbb{R}_+^{L \times K}$ and $\boldsymbol{u}_n \in \mathbb{R}_+^K$ (typically $K \ll L$). Note that the $K$ columns of $\mathbf{V}$ can be thought of as atoms of a label dictionary (or "topics" over labels) and $\boldsymbol{u}_n$ can be thought of as the atom weights or *embedding* of the label vector $\boldsymbol{y}_n$ (or "topic proportions", i.e., how active each of the $K$ topics is for example $n$). Also note that Eq. (1)-(3) can be combined as

$$\boldsymbol{y}_n = f(\boldsymbol{\lambda}_n) = f(\mathbf{V}\boldsymbol{u}_n) \tag{4}$$

where $f$ jointly denotes drawing the latent counts $\boldsymbol{m}_n$ from a Poisson (Eq. 2) with rate $\boldsymbol{\lambda}_n = \mathbf{V}\boldsymbol{u}_n$, followed by thresholding $\boldsymbol{m}_n$ at 1 (Eq. 1). In particular, note that marginalizing out $\boldsymbol{m}_n$ from Eq. 1 leads to $\boldsymbol{y}_n \sim \text{Bernoulli}(1 - \exp(-\boldsymbol{\lambda}_n))$. This link function, termed as the Bernoulli-Poisson link [28, 9], has also been used recently in modeling relational data with binary observations.

In Eq. (4), expressing the label vector $\boldsymbol{y}_n \in \{0,1\}^L$ in terms of $\mathbf{V}\boldsymbol{u}_n$ is equivalent to a low-rank assumption on the $L \times N$ label matrix $\mathbf{Y} = [\boldsymbol{y}_1 \ldots \boldsymbol{y}_N]$: $\mathbf{Y} = f(\mathbf{VU})$, where $\mathbf{V} = [\boldsymbol{v}_1 \ldots \boldsymbol{v}_K] \in \mathbb{R}_+^{L \times K}$ and $\mathbf{U} = [\boldsymbol{u}_1 \ldots \boldsymbol{u}_N] \in \mathbb{R}_+^{K \times N}$, which are modeled as follows

$$\boldsymbol{v}_k \sim \text{Dirichlet}(\eta \mathbf{1}_L) \tag{5}$$
$$u_{kn} \sim \text{Gamma}(r_k, p_{kn}(1 - p_{kn})^{-1}) \tag{6}$$
$$p_{kn} = \sigma(\boldsymbol{w}_k^\top \boldsymbol{x}_n) \tag{7}$$
$$\boldsymbol{w}_k \sim \mathcal{N}or(0, \Gamma) \tag{8}$$

$\sigma(z) = 1/(1 + \exp(-z))$, $\Gamma = \text{diag}(\tau_1^{-1}, \ldots, \tau_D^{-1})$, and hyperparameters $r_k, \tau_1, \ldots, \tau_D$ are given improper gamma priors. Since columns of $\mathbf{V}$ are Dirichlet drawn, they correspond to distributions (i.e., topics) over the labels. It is important to note here that the dependence of the label embedding $\boldsymbol{u}_n = \{u_{kn}\}_{k=1}^K$ on the feature vector $\boldsymbol{x}_n$ is achieved by making the scale parameter of the gamma prior on $\{u_{kn}\}_{k=1}^K$ depend on $\{p_{kn}\}_{k=1}^K$ which in turn depends on the features $\boldsymbol{x}_n$ via regression weight $\mathbf{W} = \{\boldsymbol{w}_k\}_{k=1}^K$ (Eq. 6 and 8).

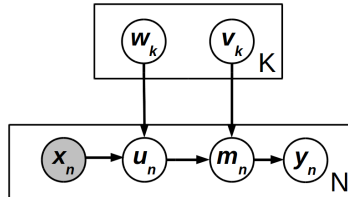

Figure 1: Graphical model for the generative process of the label vector. Hyperpriors omitted for brevity.

## 2.1 Computational scalability in the number of positive labels

For the Bernoulli-Poisson likelihood model for binary labels, we can write the conditional posterior [28, 9] of the latent count vector $\boldsymbol{m}_n$ as

$$(\boldsymbol{m}_n | \boldsymbol{y}_n, \mathbf{V}, \boldsymbol{u}_n) \sim \boldsymbol{y}_n \odot \text{Poisson}_+(\mathbf{V}\boldsymbol{u}_n) \qquad (9)$$

where $\text{Poisson}_+$ denotes the zero-truncated Poisson distribution with support only on the positive integers, and $\odot$ denotes the element-wise product. Eq. 9 suggests that the zeros in $\boldsymbol{y}_n$ will result in the corresponding elements of the latent count vector $\boldsymbol{m}_n$ being zero, almost surely (i.e., with probability one). As shown in Section 3, the sufficient statistics of the model parameters do not depend on latent counts that are equal to zero; such latent counts can be simply ignored during the inference. This aspect leads to substantial computational savings in our model, making it scale only in the number of positive labels in the label matrix. In the rest of the exposition, we will refer to our model as **BMLPL** to denote **B**ayesian **M**ulti-label **L**earning via **P**ositive **L**abels.

## 2.2 Asymmetric Link Function

In addition to the computational advantage (i.e., scaling in the number of non-zeros in the label matrix), another appealing aspect of our multi-label learning framework is that the Bernoulli-Poisson likelihood is also a more realistic model for highly sparse binary data as compared to the commonly used logistic/probit likelihood. To see this, note that the Bernoulli-Poisson model defines the probability of an observation $y$ being one as $p(y = 1|\lambda) = 1 - \exp(-\lambda)$ where $\lambda$ is the positive rate parameter. For a positive $\lambda$ on the $X$ axis, the rate of growth of the plot of $p(y = 1|\lambda)$ on the $Y$ axis from 0.5 to 1 is much slower than the rate it drops from 0.5 to 0. This behavior of the Bernoulli-Poisson link will encourage a much fewer number of nonzeros in the observed data as compared to the number of zeros. On the other hand, a logistic and probit approach both 0 and 1 at the same rate, and therefore cannot model the sparsity/skewness of the label matrix like the Bernoulli-Poisson link. Therefore, in contrast to multilabel learning models based on logistic/probit likelihood function or standard loss functions such as the hinge-loss [25, 14] for the binary labels, our proposed model provides better robustness against label imbalance.

## 3 Inference

A key aspect of our framework is that the conditional posteriors of all the model parameters are available in closed form using data augmentation strategies that we will describe below. In particular, since we model binary label matrix as thresholded counts, we are also able to leverage some of the inference methods proposed for Bayesian matrix factorization of count-valued data [27] to derive an efficient Gibbs sampler for our model.

Inference in our model requires estimating $\mathbf{V} \in \mathbb{R}_+^{L \times K}$, $\mathbf{W} \in \mathbb{R}^{D \times K}$, $\mathbf{U} \in \mathbb{R}_+^{K \times N}$, and the hyperparameters of the model. As we will see below, the latent count vectors $\{\boldsymbol{m}_n\}_{n=1}^N$ (which are functions of $\mathbf{V}$ and $\mathbf{U}$) provide sufficient statistics for the model parameters. Each element of $\boldsymbol{m}_n$ (*if* the corresponding element in $\boldsymbol{y}_n$ is one) is drawn from a truncated Poisson distribution

$$m_{ln} \sim \text{Poisson}_+(\mathbf{V}_{l,:}\boldsymbol{u}_n) = \text{Poisson}_+(\lambda_{ln}) \qquad (10)$$

$\mathbf{V}_{l,:}$ denotes the $l^{th}$ row of $\mathbf{V}$ and $\lambda_{ln} = \sum_{k=1}^K \lambda_{kln} = \sum_{k=1}^K v_{lk}u_{kn}$. Thus we can also write $m_{ln} = \sum_{k=1}^K m_{lkn}$ where $m_{lkn} \sim \text{Poisson}_+(\lambda_{kln}) = \text{Poisson}_+(v_{lk}u_{kn})$.

On the other hand, if $y_{ln} = 0$ then $m_{ln} = 0$ with probability one (Eq. (9)), and therefore need not be sampled because it does not affect the sufficient statistics of the model parameters.

Using the equivalence of Poisson and multinomial distribution [27], we can express the decomposition $m_{ln} = \sum_{k=1}^K m_{lkn}$ as a draw from a multinomial

$$[m_{l1n}, \ldots, m_{lKn}] \sim \mathcal{M}ult(m_{ln}; \zeta_{l1n}, \ldots, \zeta_{lKn}) \qquad (11)$$

where $\zeta_{lkn} = \frac{v_{lk}u_{kn}}{\sum_{k=1}^K v_{lk}u_{kn}}$. This allows us to exploit the Dirichlet-multinomial conjugacy and helps designing efficient Gibbs sampling and EM algorithms for doing inference in our model. As discussed before, the computational cost of both algorithms scales in the number of ones in the label matrix $\mathbf{Y}$, which males our model especially appealing for dealing with multilabel learning problems where the label matrix is massive but highly sparse.

## 3.1 Gibbs Sampling

Gibbs sampling for our model proceeds as follows

**Sampling V:** Using Eq. 11 and the Dirichlet-multinomial conjugacy, each column of $\mathbf{V} \in \mathbb{R}_+^{L \times K}$ can be sampled as

$$\boldsymbol{v}_k \sim \text{Dirichlet}(\eta + m_{1k}, \ldots, \eta + m_{Lk}) \tag{12}$$

where $m_{lk} = \sum_n m_{lnk}, \forall l = 1, \ldots, L$.

**Sampling U:** Using the gamma-Poisson conjugacy, each entry of $\mathbf{U} \in \mathbb{R}_+^{K \times N}$ can be sampled as

$$u_{kn} \sim \text{Gamma}(r_k + m_{kn}, p_{kn}) \tag{13}$$

where $m_{kn} = \sum_l m_{lnk}$ and $p_{kn} = \sigma(\boldsymbol{w}_k^\top \boldsymbol{x}_n)$.

**Sampling W:** Since $m_{kn} = \sum_l m_{lnk}$ and $m_{lnk} \sim \text{Poisson}_+(v_{lk} u_{kn})$, $p(m_{kn}|u_{kn})$ is also Poisson. Further, since $p(u_{kn}|r, p_{kn})$ is gamma, we can integrate out $u_{kn}$ from $p(m_{kn}|u_{kn})$ which gives

$$m_{kn} = \text{NegBin}(r_k, p_{kn})$$

where $\text{NegBin}(.,.)$ denotes the negative Binomial distribution.

Although the negative Binomial is not conjugate to the Gaussian prior on $\boldsymbol{w}_k$, we leverage the Pólya-Gamma strategy [17] data augmentation to "Gaussianify" the negative Binomial likelihood. Doing this, we are able to derive closed form Gibbs sampling updates $\boldsymbol{w}_k$, $k = 1, \ldots, K$. The Pólya-Gamma (PG) strategy is based on sampling a set of auxiliary variables, one for each observation (which, in the context of sampling $\boldsymbol{w}_k$, are the latent counts $m_{kn}$). For sampling $\boldsymbol{w}_k$, we draw $N$ Pólya-Gamma random variables [17] $\omega_{k1}, \ldots, \omega_{kN}$ (one for each training example) as

$$\omega_{kn} \sim \text{PG}(m_{kn} + r_k, \boldsymbol{w}_k^\top \boldsymbol{x}_n) \tag{14}$$

where $\text{PG}(.,.)$ denotes the Pólya-Gamma distribution [17].

Given these PG variables, the posterior distribution of $\boldsymbol{w}_k$ is Gaussian $\mathcal{N}or(\mu_{w_k}, \Sigma_{w_k})$ where

$$\Sigma_{w_k} = (\mathbf{X}\Omega_k\mathbf{X}^\top + \Gamma^{-1})^{-1} \tag{15}$$
$$\mu_{w_k} = \Sigma_{w_k}\mathbf{X}\kappa_k \tag{16}$$

where $\Omega_k = \text{diag}(\omega_{k1}, \ldots, \omega_{kN})$ and $\kappa_k = [(m_{k1} - r_k)/2, \ldots, (m_{kN} - r_k)/2]^\top$.

**Sampling the hyperparameters:** The hyperparameter $r_k$ is given a gamma prior and can be sampled easily. The other hyperparameters $\tau_1, \ldots, \tau_D$ are estimated using Type-II maximum likelihood estimation [22].

## 3.2 Expectation Maximization

The Gibbs sampler described in Section 3.1 is efficient and has a computational complexity that scales in the number of ones in the label matrix. To further scale up the inference, we also develop an efficient Expectation-Maximization (EM) inference algorithm for our model. In the E-step, we need to compute the expectations of the local variables $\mathbf{U}$, the latent counts, and the Pólya-Gamma variables $\omega_{k1}, \ldots, \omega_{kN}$, for $k = 1, \ldots, K$. These expectations are available in closed form and can thus easily be computed. In particular, the expectation of each Pólya-Gamma variable $\omega_{kn}$ is very efficient to compute and is available in closed form [20]

$$\mathbb{E}[\omega_{kn}] = \frac{(m_{kn} + r_k)}{2\boldsymbol{w}_k^\top \boldsymbol{x}_n} \tanh(\boldsymbol{w}_k^\top \boldsymbol{x}_n/2) \tag{17}$$

The M-step involves a maximization w.r.t. $\mathbf{V}$ and $\mathbf{W}$, which essentially involves solving for their *maximum-a-posteriori* (MAP) estimates, which are available in closed form. In particular, as shown in [20], estimating $\boldsymbol{w}_k$ requires solving a linear system which, in our case, is of the form

$$\mathbf{S}_k \boldsymbol{w}_k = \boldsymbol{d}_k \tag{18}$$

where $\mathbf{S}_k = \mathbf{X}\Omega_k\mathbf{X}^\top + \Gamma^{-1}$, $\boldsymbol{d}_k = \mathbf{X}\kappa_k$, $\Omega_k$ and $\kappa_k$ are defined as in Section 3.1, except that the Pólya-Gamma random variables are replaced by their expectations given by Eq. 17. Note that Eq. 18

can be straighforwardly solved as $\boldsymbol{w}_k = \mathbf{S}_k^{-1}\boldsymbol{d}_k$. However, convergence of the EM algorithm [20] does not require solving for $\boldsymbol{w}_k$ exactly in each EM iteration and running a couple of iterations of any of the various iterative methods that solves a linear system of equations can be used for this step. We use the Conjugate Gradient [2] method to solve this, which also allows us to exploit the sparsity in $\mathbf{X}$ and $\Omega_k$ to very efficiently solve this system of equations, even when $D$ and $N$ are very large. Although in this paper, we only use the batch EM, it is possible to speed it up even further using an online version of this EM algorithm, as shown in [20]. The online EM processes data in small minibatches and in each EM iteration updates the sufficient statistics of the global parameters. In our case, these sufficient statistics include $\mathbf{S}_k$ and $\boldsymbol{d}_k$, for $k = 1, \ldots, K$, and can be updated as

$$
\begin{aligned}
\mathbf{S}_k^{(t+1)} &= (1 - \gamma_t)\mathbf{S}_k^{(t)} + \gamma_t \mathbf{X}^{(t)}\Omega_k^{(t)}\mathbf{X}^{(t)^\top} \\
\boldsymbol{d}_k^{(t+1)} &= (1 - \gamma_t)\boldsymbol{d}_k^{(t)} + \gamma_t \mathbf{X}^{(t)}\kappa_k^{(t)}
\end{aligned}
$$

where $\mathbf{X}^{(t)}$ denotes the set of examples in the current minibatch, and $\Omega_k^{(t)}$ and $\kappa_k^{(t)}$ denote quantities that are computed using the data from the current minibatch.

### 3.3 Predicting Labels for Test Examples

Predicting the label vector $\boldsymbol{y}_* \in \{0, 1\}^L$ for a new test example $\boldsymbol{x}_* \in \mathbb{R}^D$ can be done as

$$
p(\boldsymbol{y}_* = 1|\boldsymbol{x}_*) = \int_{\boldsymbol{u}_*} (1 - \exp(-\mathbf{V}\boldsymbol{u}_*))p(\boldsymbol{u}_*)d\boldsymbol{u}_*
$$

If using Gibbs sampling, the integral above can be approximated using samples $\{\boldsymbol{u}_*^{(m)}\}_{m=1}^M$ from the posterior of $\boldsymbol{u}_*$. It is also possible to integrate out $\boldsymbol{u}_*$ (details skipped for brevity) and get closed form estimates of probability of each label $y_{l*}$ in terms of the model parameters $\mathbf{V}$ and $\mathbf{W}$, and it is given by

$$
p(y_{l*} = 1|\boldsymbol{x}_*) = 1 - \prod_{k=1}^K \frac{1}{[V_{lk}\exp(\boldsymbol{w}_k^\top\boldsymbol{x}_*) + 1]^{r_k}} \tag{19}
$$

## 4 Computational Cost

Computing the latent count $m_{ln}$ for each nonzero entry $y_{ln}$ in $\mathbf{Y}$ requires computing $[m_{l1n}, \ldots, m_{lKn}]$, which takes $O(K)$ time; therefore computing all the latent counts takes $O(\mathrm{nnz}(\mathbf{Y})K)$ time, which is very efficient if $\mathbf{Y}$ has very few nonzeros (which is true of most real-world multi-label learning problems). Estimating $\mathbf{V}$, $\mathbf{U}$, and the hyperparameters is relatively cheap and can be done very efficiently. The Pólya-Gamma variables, when doing Gibbs sampling, can be efficiently sampled using methods described in [17]; and when doing EM, these can be even more cheaply computed because the Pólya-Gamma expectations, which are available in closed form (as a hyperbolic tan function), can be very efficiently computed [20]. The most dominant step is estimating $\mathbf{W}$; when doing Gibbs sampling, if done *naïvely*, it would $O(DK^3)$ time if sampling $\mathbf{W}$ row-wise, and $O(KD^3)$ time if sampling column-wise. However, if using the EM algorithm, estimating $\mathbf{W}$ can be done much more efficiently, e.g., using Conjugate Gradient updates because, it is not even required to solved for $\mathbf{W}$ *exactly* in each iteration of the EM algorithm [20]. Also note that since most of the parameters updates for different $k = 1, \ldots, K, n = 1, \ldots, N$ are all independent of each other, our Gibbs sampler and the EM algorithms can be easily parallelized/block-updated.

## 5 Connection: Topic Models with Meta-Data

As discussed earlier, our multi-label learning framework is similar in spirit to a topic model as the label embeddings naturally correspond to topics - each Dirichlet-drawn column $\boldsymbol{v}_k$ of the matrix $\mathbf{V} \in \mathbb{R}_+^{L \times K}$ can be seen as representing a "topic". In fact, our model, interestingly, can directly be seen as a topic model [3, 27] where we have side-information associated with each document (e.g., document features). For example, if each document $\boldsymbol{y}_n \in \{0, 1\}^L$ (in a bag-of-words representation with vocabulary of size $L$) may also have some meta-data $\boldsymbol{x}_n \in \mathbb{R}^D$ associated with it. Our model can therefore also be used to perform topic modeling of text documents with such meta-data [15, 12, 29, 19] in a robust and scalable manner.

# 6 Related Work

Despite a significant number of methods proposed in the recent years, learning from multi-label data continues to remain an active area of research, especially due to the recent surge of interest in learning when the output space (i.e., the number of labels) is massive. To handle the huge dimensionality of the label space, a common approach is to embed the labels in a lower-dimensional space, e.g., using methods such as Canonical Correlation Analysis or other methods for jointly embedding feature and label vectors [26, 5, 23], Compressed Sensing[8, 10], or by assuming that the matrix consisting of the weight vectors of all the labels is a low-rank matrix [25]. Another interesting line of work on label embedding methods makes use of random projections to reduce the label space dimensionality [11, 16], or use methods such as multitask learning (each label is a task).

Our proposed framework is most similar in spirit to the aforementioned class of label embedding based methods (we compare with some of these in our experiments). In contrast to these methods, our framework reduces the label-space dimensionality via a *nonlinear* mapping (Section 2), our framework has accompanying inference algorithms that scale in the number of positive labels 2.1, has an underlying generative model that more realistically models the imbalanced nature of the labels in the label matrix (Section 2.2), can deal with missing labels, and is easily parallelizable. Also, the connection to topic models provide a nice interpretability to the results, which is usually not possible with the other methods (e.g., in our model, the columns of the matrix $\mathbf{V}$ can be seen as a set of topics over the labels; in Section 7.2, we show an experiment on this). Moreover, although in this paper, we have focused on the multi-label learning problem, our framework can also be applied for *multiclass* problems via the one-vs-all reduction, in which case the label matrix is usually very sparse (each column of the label matrix represents the labels of a single one-vs-all binary classification problem).

Finally, although not a focus of this paper, some other important aspects of the multi-label learning problem have also been looked at in recent work. For example, fast prediction at test time is an important concern when the label space is massive. To deal with this, some recent work focuses on methods that only incur a logarithmic cost (in the number of labels) at test time [1, 18], e.g., by inferring and leveraging a tree structure over the labels.

# 7 Experiments

We evaluate the proposed multi-label learning framework on four benchmark multi-label data sets - bibtex, delicious, compphys, eurlex [25], with their statistics summarized in Table 1. The data sets we use in our experiments have both feature and label dimensions that range from a few hundreds to a several thousands. In addition, the feature and/or label matrices are also quite sparse.

| Data set | $D$ | $L$ | Training set | | | Test set | | |
|---|---|---|---|---|---|---|---|---|
| | | | $N_{train}$ | $\bar{L}$ | $\bar{D}$ | $N_{test}$ | $\bar{L}$ | $\bar{D}$ |
| bibtex | 1836 | 159 | 4880 | 2.40 | 68.74 | 2515 | 2.40 | 68.50 |
| delicious | 500 | 983 | 12920 | 19.03 | 18.17 | 3185 | 19.00 | 18.80 |
| compphys | 33,284 | 208 | 161 | 9.80 | 792.78 | 40 | 11.83 | 899.20 |
| eurlex | 5000 | 3993 | 17413 | 5.30 | 236.69 | 1935 | 5.32 | 240.96 |

Table 1: Statistics of the data sets used in our experiments. $\bar{L}$ denotes average number of positive labels per example; $\bar{D}$ denotes the average number of nonzero features per example.

We compare the proposed model **BMLPL** with four state-of-the-art methods. All these methods, just like our method, are based on the assumption that the label vectors live in a low dimensional space.

- CPLST: Conditional Principal Label Space Transformation [5]: CPLST is based on embedding the label vectors conditioned on the features.
- BCS: Bayesian Compressed Sensing for multi-label learning [10]: BCS is a Bayesian method that uses the idea of doing compressed sensing on the labels [8].
- WSABIE: It assumes that the feature as well as the label vectors live in a low dimensional space. The model is based on optimizing a weighted approximate ranking loss [23].
- LEML: Low rank Empirical risk minimization for multi-label learning [25]. For LEML, we report the best results across the three loss functions (squared, logistic, hinge) they propose.

Table 2 shows the results where we report the Area Under the ROC Curve (AUC) for each method on all the data sets. For each method, as done in [25], we vary the label space dimensionality from 20% - 100% of $L$, and report the best results. For BMLPL, both Gibbs sampling and EM based inference perform comparably (though EM runs much faster than Gibbs); here we report results obtained with EM inference only (Section 7.4 provides another comparison between these two inference methods). The EM algorithms were run for 1000 iterations and they converged in all the cases.

As shown in the results in Table 2, in almost all of the cases, the proposed BMLPL model performs better than the other methods (except for compphys data sets where the AUC is slightly worse than LEML). The better performance of our model justifies the flexible Bayesian formulation and also shows the evidence of the robustness provided by the asymmetric link function against sparsity and label imbalance in the label matrix (note that the data sets we use have very sparse label matrices).

| | CPLST | BCS | WSABIE | LEML | BMLPL |
|---|---|---|---|---|---|
| bibtex | 0.8882 | 0.8614 | 0.9182 | 0.9040 | **0.9210** |
| delicious | 0.8834 | 0.8000 | 0.8561 | 0.8894 | **0.8950** |
| compphys | 0.7806 | 0.7884 | 0.8212 | **0.9274** | 0.9211 |
| eurlex | - | - | 0.8651 | 0.9456 | **0.9520** |

Table 2: Comparison of the various methods in terms of AUC scores on all the data sets. Note: CPLST and BCS were not feasible to run on the eurlex data, so we are unable to report those numbers here.

## 7.1 Results with Missing Labels

Our generative model for the label matrix can also handle missing labels (the missing labels may include both zeros or ones). We perform an experiment on two of the data sets - bibtex and compphys - where only 20% of the labels from the label matrix are revealed (note that, of all these revealed labels, our model uses only the positive labels), and compare our model with LEML and BCS (both are capable of handling missing labels). The results are shown in Table 3. For each method, we set $K = 0.4L$. As the results show, our model yields better results as compared to the competing methods even in the presence of missing labels.

| | BCS | LEML | BMLPL |
|---|---|---|---|
| bibtex | 0.7871 | 0.8332 | **0.8420** |
| compphys | 0.6442 | 0.7964 | **0.8012** |

Table 3: AUC scores with only 20% labels observed.

## 7.2 Qualitative Analysis: Topic Modeling on Eurlex Data

Since in our model, each column of the $L \times K$ matrix $\mathbf{V}$ represents a distribution (i.e., a "topic") over the labels, to assess its ability of discovering meaningful topics, we run an experiment on the Eurlex data with $K = 20$ and look at each column of $\mathbf{V}$. The Eurlex data consists of 3993 labels (each of which is a tags; a document can have a subset of the tags), so each column in $\mathbf{V}$ is of that size. In Table 4, we show five of the topics (and top five labels in each topic, based on the magnitude of the entries in the corresponding column of $\mathbf{V}$). As shown in Table 4, our model is able to discover clear and meaningful topics from the Eurlex data, which shows its usefulness as a topic model when each document $\boldsymbol{y}_n \in \{0, 1\}^L$ has features in form of meta data $\boldsymbol{x}_n \in \mathbb{R}^D$ associated with it.

| Topic 1 (Nuclear) | Topic 2 (Agreements) | Topic 3 (Environment) | Topic 4 (Stats & Data) | Topic 5 (Fishing Trade) |
|---|---|---|---|---|
| nuclear safety | EC agreement | environmental protection | community statistics | fishing regulations |
| nuclear power station | trade agreement | waste management | statistical method | fishing agreement |
| radioactive effluent | EC interim agreement | env. monitoring | agri. statistics | fishery management |
| radioactive waste | trade cooperation | dangerous substance | statistics | fishing area |
| radioactive pollution | EC coop. agree. | pollution control measures | data transmission | conservation of fish stocks |

Table 4: Most probable words in different topics.

### 7.3 Scalability w.r.t. Number of Positive Labels

To demonstrate the linear scalability in the number of positive labels, we run an experiment on the Delicious data set by varying the number of positive labels used for training the model from 20% to 100% (to simulate this, we simply treat all the other labels as zeros, so as to have a constant label matrix size). We run each experiment for 100 iterations (using EM for the inference) and report the running time for each case. Fig. 2 (left) shows the results which demonstrates the roughly linear scalability w.r.t. the number of positive labels. This experiment is only meant for a small illustration. Note than the actual scalability will also depend on the relative values of $D$ and $L$ and the sparsity of $\mathbf{Y}$. In any case, the amount of computations the involve the labels (both positive and negatives) only depend on the positive labels, and this part, for our model, is clearly linear in the number of positive labels in the label matrix.

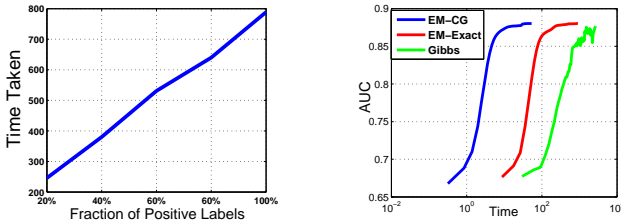

Figure 2: (Left) Scalability w.r.t. number of positive labels. (Right) Time vs accuracy comparison for Gibbs and EM (with exact and with CG based M steps)

### 7.4 Gibbs Sampling vs EM

We finally show another experiment comparing both Gibbs sampling and EM for our model in terms of accuracy vs running time. We run each inference method only for 100 iterations. For EM, we try two settings: EM with an *exact* M step for $\mathbf{W}$, and EM with an *approximate* M step where we run 2 steps of conjugate gradient (CG). Fig. 2 (right), shows a plot comparing each inference method in terms of the accuracy vs running time. As Fig. 2 (right) shows, the EM algorithms (both exact as well as the one that uses CG) attain reasonably high AUC scores in a short amount of time, which the Gibbs sampling takes much longer per iteration and seems to converge rather slowly. Moreover, remarkably, EM with 2 iterations CG in each $M$ steps seems to perform comparably to the EM with an exact $M$ step, while running considerably faster. As for the Gibbs sampler, although it runs slower than the EM based inference, it should be noted that the Gibbs sampler would still be considerably faster than other fully Bayesian methods for multi-label prediction (such as BCS [10]) because it only requires evaluating the likelihoods over the positive labels in the label matrix. Moreover, the step involving sampling of the $\mathbf{W}$ matrix can be made more efficient by using cholesky decompositions which can avoid matrix inversions needed for computing the covariance of the Gaussian posterior on $\boldsymbol{w}_k$.

## 8 Discussion and Conclusion

We have presented a scalable Bayesian framework for multi-label learning. In addition to providing a flexible model for sparse label matrices, our framework is also computationally attractive and can scale to massive data sets. The model is easy to implement and easy to parallelize. Both full Bayesian inference via simple Gibbs sampling and EM based inference can be carried out in this model in a computationally efficient way. Possible future work includes developing online Gibbs and online EM algorithms to further enhance the scalability of the proposed framework to handle even bigger data sets. Another possible extension could be to additionally impose label correlations more explicitly (in addition to the low-rank structure already imposed by the current model), e.g., by replacing the Dirichlet distribution on the columns of $\mathbf{V}$ with logistic normal distributions [4]. Because our framework allows efficiently computing the predictive distribution of the labels (as shown in Section 3.3), it can be easily extend for doing active learning on the labels [10]. Finally, although here we only focused on multi-label learning, our framework can be readily used as a robust and scalable alternative to methods that perform binary matrix factorization with side-information.

**Acknowledgements** This research was supported in part by ARO, DARPA, DOE, NGA and ONR

# References

[1] Rahul Agrawal, Archit Gupta, Yashoteja Prabhu, and Manik Varma. Multi-label learning with millions of labels: Recommending advertiser bid phrases for web pages. In *WWW*, 2013.

[2] Dimitri P Bertsekas. *Nonlinear programming*. Athena scientific Belmont, 1999.

[3] David M Blei, Andrew Y Ng, and Michael I Jordan. Latent dirichlet allocation. *JMLR*, 2003.

[4] Jianfei Chen, Jun Zhu, Zi Wang, Xun Zheng, and Bo Zhang. Scalable inference for logistic-normal topic models. In *NIPS*, 2013.

[5] Yao-Nan Chen and Hsuan-Tien Lin. Feature-aware label space dimension reduction for multi-label classification. In *NIPS*, 2012.

[6] Eva Gibaja and Sebastián Ventura. Multilabel learning: A review of the state of the art and ongoing research. *Wiley Interdisciplinary Reviews: Data Mining and Knowledge Discovery*, 2014.

[7] Eva Gibaja and Sebastián Ventura. A tutorial on multilabel learning. *ACM Comput. Surv.*, 2015.

[8] Daniel Hsu, Sham Kakade, John Langford, and Tong Zhang. Multi-label prediction via compressed sensing. In *NIPS*, 2009.

[9] Changwei Hu, Piyush Rai, and Lawrence Carin. Zero-truncated poisson tensor factorization for massive binary tensors. In *UAI*, 2015.

[10] Ashish Kapoor, Raajay Viswanathan, and Prateek Jain. Multilabel classification using bayesian compressed sensing. In *NIPS*, 2012.

[11] Nikos Karampatziakis and Paul Mineiro. Scalable multilabel prediction via randomized methods. *arXiv preprint arXiv:1502.02710*, 2015.

[12] Dae I Kim and Erik B Sudderth. The doubly correlated nonparametric topic model. In *NIPS*, 2011.

[13] Xiangnan Kong, Zhaoming Wu, Li-Jia Li, Ruofei Zhang, Philip S Yu, Hang Wu, and Wei Fan. Large-scale multi-label learning with incomplete label assignments. In *SDM*, 2014.

[14] Xin Li, Feipeng Zhao, and Yuhong Guo. Conditional restricted boltzmann machines for multi-label learning with incomplete labels. In *AISTATS*, 2015.

[15] David Mimno and Andrew McCallum. Topic models conditioned on arbitrary features with dirichlet-multinomial regression. In *UAI*, 2008.

[16] Paul Mineiro and Nikos Karampatziakis. Fast label embeddings for extremely large output spaces. In *ICLR Workshop*, 2015.

[17] Nicholas G Polson, James G Scott, and Jesse Windle. Bayesian inference for logistic models using pólya–gamma latent variables. *Journal of the American Statistical Association*, 108(504):1339–1349, 2013.

[18] Yashoteja Prabhu and Manik Varma. FastXML: a fast, accurate and stable tree-classifier for extreme multi-label learning. In *KDD*, 2014.

[19] Maxim Rabinovich and David Blei. The inverse regression topic model. In *ICML*, 2014.

[20] James G Scott and Liang Sun. Expectation-maximization for logistic regression. *arXiv preprint arXiv:1306.0040*, 2013.

[21] Farbound Tai and Hsuan-Tien Lin. Multilabel classification with principal label space transformation. *Neural Computation*, 2012.

[22] Michael E Tipping. Bayesian inference: An introduction to principles and practice in machine learning. In *Advanced lectures on machine Learning*, pages 41–62. Springer, 2004.

[23] Jason Weston, Samy Bengio, and Nicolas Usunier. WSABIE: Scaling up to large vocabulary image annotation. In *IJCAI*, 2011.

[24] Yan Yan, Glenn Fung, Jennifer G Dy, and Romer Rosales. Medical coding classification by leveraging inter-code relationships. In *KDD*, 2010.

[25] Hsiang-Fu Yu, Prateek Jain, Purushottam Kar, and Inderjit S Dhillon. Large-scale multi-label learning with missing labels. In *ICML*, 2014.

[26] Yi Zhang and Jeff G Schneider. Multi-label output codes using canonical correlation analysis. In *AISTATS*, 2011.

[27] M. Zhou, L. A. Hannah, D. Dunson, and L. Carin. Beta-negative binomial process and poisson factor analysis. In *AISTATS*, 2012.

[28] Mingyuan Zhou. Infinite edge partition models for overlapping community detection and link prediction. In *AISTATS*, 2015.

[29] Jun Zhu, Ni Lao, Ning Chen, and Eric P Xing. Conditional topical coding: an efficient topic model conditioned on rich features. In *KDD*, 2011.

